# Topology Constraints in Graphical Models

**Marcelo Fiori**
Universidad de la
República, Uruguay
mfiori@fing.edu.uy

**Pablo Musé**
Universidad de la
República, Uruguay
pmuse@fing.edu.uy

**Guillermo Sapiro**
Duke University
Durham, NC 27708
guillermo.sapiro@duke.edu

## Abstract

Graphical models are a very useful tool to describe and understand natural phenomena, from gene expression to climate change and social interactions. The topological structure of these graphs/networks is a fundamental part of the analysis, and in many cases the main goal of the study. However, little work has been done on incorporating prior topological knowledge onto the estimation of the underlying graphical models from sample data. In this work we propose extensions to the basic joint regression model for network estimation, which explicitly incorporate graph-topological constraints into the corresponding optimization approach. The first proposed extension includes an eigenvector centrality constraint, thereby promoting this important prior topological property. The second developed extension promotes the formation of certain motifs, triangle-shaped ones in particular, which are known to exist for example in genetic regulatory networks. The presentation of the underlying formulations, which serve as examples of the introduction of topological constraints in network estimation, is complemented with examples in diverse datasets demonstrating the importance of incorporating such critical prior knowledge.

## 1 Introduction

The estimation of the inverse of the covariance matrix (also referred to as *precision matrix* or *concentration matrix*) is a very important problem with applications in a number of fields, from biology to social sciences, and is a fundamental step in the estimation of underlying data networks. The *covariance selection* problem, as introduced by Dempster (1972), consists in identifying the zero pattern of the precision matrix. Let $X = (X_1 \dots X_p)$ be a p-dimensional multivariate normal distributed variable, $X \sim \mathcal{N}(0, \Sigma)$, and $C = \Sigma^{-1}$ its concentration matrix. Then two coordinates $X_i$ and $X_j$ are conditionally independent given the other variables if and only if $C(i, j) = 0$ (Lauritzen, 1996). This property motivates the representation of the conditional dependency structure in terms of a graphical model $G = (V, E)$, where the set of nodes $V$ corresponds to the $p$ coordinates and the edges $E$ represent conditional dependency. Note that the zero pattern of the $G$ adjacency matrix coincides with the zero pattern of the concentration matrix. Therefore, the estimation of this graph $G$ from $k$ random samples of $X$ is equivalent to the covariance selection problem. The estimation of $G$ using $\ell_1$ (sparsity promoting) optimization techniques has become very popular in recent years.

This estimation problem becomes particularly interesting and hard at the same time when the number of samples $k$ is smaller than $p$. Several real life applications lie in this "small $k$-large $p$" setting. One of the most studied examples, and indeed with great impact, is the inference of genetic regulatory networks (GRN) from DNA microarray data, where typically the number $p$ of genes is much larger than the number $k$ of experiments. Like in the vast majority of applications, these networks have some very well known topological properties, such as sparsity (each node is connected with only a few other nodes), scale-free behavior, and the presence of hubs (nodes connected with many other vertices). All these properties are shared with many other real life networks like Internet, citation networks, and social networks (Newman, 2010).

Genetic regulatory networks also contain a small set of recurring patterns called *motifs*. The systematic presence of these motifs has been first discovered in *Escherichia coli* (Shen-Orr et al., 2002), where it was found that the frequency of these patterns is much higher than in random networks, and since then they have been identified in other organisms, from bacteria to yeast, plants and animals.

The topological analysis of networks is fundamental, and often the essence of the study. For example, the proper identification of hubs or motifs in GRN is crucial. Thus, the agreement of the reconstructed topology with the original or expected one is critical. Sparsity has been successfully exploited via $\ell_1$ penalization in order to obtain consistent estimators of the precision matrix, but little work has been done with other graph-topological properties, often resulting in the estimation of networks that lack critical known topological structures, and therefore do not look natural. Incorporating such topological knowledge in network estimation is the main goal of this work.

*Eigenvector centrality* (see Section 3 for the precise definition) is a well-known measure of the importance and the connectivity of each node, and typical centrality distributions are known (or can be estimated) for several types of networks. Therefore, we first propose to incorporate this structural information into the optimization procedure for network estimation in order to control the topology of the resulting network. This centrality constraint is useful when some prior information about the graphical model is known, for example, in dynamic networks, where the topology information of the past can be used; in networks which we know are similar to other previously studied graphs; or in networks that model a physical phenomenon for which a certain structure is expected.

As mentioned, it has been observed that genetic regulatory networks are conformed by a few geometric patterns, repeated several times. One of these motifs is the so-called *feedforward loop*, which is manifested as a triangle in the graph. Although it is thought that these important motifs may help to understand more complex organisms, no effort has been made to include this prior information in the network estimation problem. As a second example of the introduction of topological constraints, we propose a simple modification to the $\ell_1$ penalty, weighting the edges according to their local structure, in order to favor the appearance of these motifs in the estimated network.

Both developed extensions here presented are very flexible, and they can be combined with each other or with other extensions reported in literature.

To recapitulate, we propose several contributions to the network estimation problem: we show the importance of adding topological constraints; we propose an extension to $\ell_1$ models in order to impose the eigenvector centrality; we show how to transfer topology from one graph to another; we show that even with the centrality estimated from the same data, the proposed extension outperforms the basic model; we present a weighting modification to the $\ell_1$ penalty favoring the appearance of motifs; as illustrative examples, we show how the proposed framework improves the edge and motif detection in the *E. coli* network, and how the approach is important as well in financial applications.

The rest of this paper is organized as follows. In Section 2 we describe the basic precision matrix estimation models used in this work. In Section 3 we introduce the eigenvector centrality and describe how to impose it in graph estimation. We propose the weighting method for motifs estimation in Section 4. Experimental results are presented in Section 5, and we conclude in Section 6.

## 2   Graphical Model Estimation

Let $\mathbf{X}$ be a $k \times p$ matrix containing $k$ independent observations of $X$, and let us denote by $\mathbf{X}_i$ the $i$-th column of $\mathbf{X}$. Two main families of approaches use sparsity constraints when inferring the structure of the precision matrix. The first one is based on the fact that the $(i, j)$ element of $\Sigma^{-1}$ is, up to a constant, the regression coefficient $\beta_j^i$ in $\mathbf{X}_i = \sum_{l \neq i} \beta_l^i \mathbf{X}_l + \varepsilon_i$, where $\varepsilon_i$ is uncorrelated with $\{\mathbf{X}_l | l \neq i\}$. Following this property, the neighborhood selection technique by Meinshausen & Bühlmann (2006) consists in solving $p$ independent $\ell_1$ regularized problems (Tibshirani, 1996),

$$\arg \min_{\boldsymbol{\beta}^i : \beta_i^i = 0} \frac{1}{k} ||\mathbf{X}_i - \mathbf{X}\boldsymbol{\beta}^i||^2 + \lambda ||\boldsymbol{\beta}^i||_1 \,,$$

where $\boldsymbol{\beta}^i$ is the vector of $\beta_j^i$s. While this is an asymptotically consistent estimator of the $\Sigma^{-1}$ zero pattern, $\beta_j^i$ and $\beta_i^j$ are not necessarily equal since they are estimated independently. Peng et al. (2009) propose a joint regression model which guarantees symmetry. This regression of the form $\mathbf{X} \approx \mathbf{X}B$, with $B$ sparse, symmetric, and with null diagonal, allows to control the topology of the graph defined by the non-zero pattern of $B$, as it will be later exploited in this work. Friedman

et al. (2010) also solve a symmetric version of the model by Meinshausen & Bühlmann (2006) and incorporate some structure penalties as the grouped lasso by Yuan & Lin (2006).

Methods of the second family are based on a maximum likelihood (ML) estimator with an $\ell_1$ penalty (Yuan & Lin, 2007; Banerjee et al., 2008; Friedman et al., 2008). Specifically, if $S$ denotes the empirical covariance matrix, the solution is the matrix $\Theta$ which solves the optimization problem

$$\max_{\Theta \succ 0} \ \log \det \Theta - \text{tr}(S\Theta) - \lambda \sum_{i,j} |\Theta_{ij}| \ .$$

An example of an extension to both models (the regression and ML approaches), and the first to explicitly consider additional classical network properties, is the work by Liu & Ihler (2011), which modifies the $\ell_1$ penalty to derive a non-convex optimization problem that favors scale-free networks.

A completely different technique for network estimation is the use of the PC-Algorithm to infer acyclic graphs (Kalisch & Bühlmann, 2007). This method starts from a complete graph and recursively deletes edges according to conditional independence decisions. In this work, we use this technique to estimate the graph eigenvector centrality.

## 3 Eigenvector Centrality Model Extension

Node degree (the number of connections of a node) is the simplest algebraic property than can be defined over a graph, but it is very local as it only takes into account the neighborhood of the node. A more global measure of the node importance is the so-called *centrality*, in any of its different variants. In this work, we consider the *eigenvector centrality*, defined as the dominant eigenvector (the one corresponding to the largest eigenvalue) of the corresponding network connectivity matrix. The coordinates of this vector (which are all non-negatives) are the corresponding centrality of each node, and provide a measure of the influence of the node in the network (Google's *PageRank* is a variant of this centrality measure). Distributions of the eigenvector centrality values are well known for a number of graphs, including scale-free networks as the Internet and GRN (Newman, 2010).

In certain situations, we may have at our disposal an estimate of the centrality vector of the network to infer. This may happen, for instance, because we already had preliminary data, or we know a network expected to be similar, or simply someone provided us with some partial information about the graph structure. In those cases, we would like to make use of this important side information, both to improve the overall network estimation and to guarantee that the inferred graph is consistent with our prior topological knowledge. In what follows we propose an extension of the joint regression model which is capable of controlling this topological property of the estimated graph.

To begin with, let us remark that as $\Sigma$ is positive-semidefinite and symmetric, all its eigenvalues are non-negative, and thus so are the eigenvalues of $\Sigma^{-1}$. By virtue of the Perron-Frobenius Theorem, for any adjacency matrix $A$, the eigenvalue with largest absolute value is positive. Therefore for precision and graph connectivity matrices it holds that $\max_{||v||=1} |\langle Av, v \rangle| = \max_{||v||=1} \langle Av, v \rangle$, and moreover, the eigenvector centrality is $c = \arg \max_{||v||=1} \langle Av, v \rangle$.

Suppose that we know an estimate of the centrality $c \in \mathbb{R}^p$, and want the inferred network to have centrality close to it. We start from the basic joint regression model,

$$\min_{B} ||\mathbf{X} - \mathbf{X}B||_F^2 + \lambda_1 ||B||_{\ell_1} \ , \qquad s.t. \ B \text{ symmetric, } B_{ii} = 0 \ \forall i, \tag{1}$$

and add the centrality penalty,

$$\min_{B} ||\mathbf{X} - \mathbf{X}B||_F^2 + \lambda_1 ||B||_{\ell_1} - \lambda_2 \langle Bc, c \rangle \ , \qquad s.t. \ B \text{ symmetric, } B_{ii} = 0 \ \forall i \tag{2}$$

where $|| \cdot ||_F$ is the Frobenius norm and $||B||_{\ell_1} = \sum_{i,j} |B_{ij}|$. The minus sign is due to the minimization instead of maximization, and since the term $\langle Bc, c \rangle$ is linear, the problem is still convex.

Although $B$ is intended to be a good estimation of the precision matrix (up to constants), formulations (1) or (2) do not guarantee that $B$ will be positive-semidefinite, and therefore the leading eigenvalue might not be positive. One way to address this is to add the positive-semidefinite constraint in the formulation, which keeps the problem convex. However, in all of our experiments with model (2) the spectral radius resulted positive, so we decided to use this simpler formulation due to the power of the available solvers.

Note that we are imposing the dominant eigenvector of the graph connectivity matrix $A$ to a non-binary matrix $B$. We have exhaustive empirical evidence that the leading eigenvector of the matrix

$B$ obtained by solving (2), and the leading eigenvector corresponding to the resulting connectivity matrix (the binarization of $B$) are very similar (see Section 5.1). In addition, based on Wolf & Shashua (2005), these type of results can be proved theoretically (Zeitouni, 2012).

As shown in Section 5, when the correct centrality is imposed, our proposed model outperforms the joint regression model, both in correct reconstructed edge rates and topology. This is still true when we only have a noisy version of $c$. Even if we do not have prior information at all, and we estimate the centrality from the data with a pre-run of the PC-Algorithm, we obtain improved results.

The model extension here presented is general, and the term $\langle Bc, c \rangle$ can be included in maximum likelihood based approaches like Banerjee et al. (2008); Friedman et al. (2008); Yuan & Lin (2007).

### 3.1 Implementation

Following Peng et al. (2009), the matrix optimization (2) can be cast as a classical vector $\ell_1$ penalty problem. The symmetry and null diagonal constraints are handled considering only the upper triangular sub-matrix of $B$ (excluding the diagonal), and forming a vector $\theta$ with its entries: $\theta = (B_{12}, B_{13}, \ldots, B_{(p-1)p})$. Let us consider a $pk \times 1$ column vector $\mathbf{y}$ formed by concatenating all the columns of $\mathbf{X}$. It is easy to find a $pk \times p(p-1)/2$ matrix $\mathbf{X_t}$ such that $||\mathbf{X} - \mathbf{X}B||_F^2 = ||\mathbf{y} - \mathbf{X_t}\theta||_2^2$ (see Peng et al. (2009) for details), and trivially $||B||_{\ell_1} = 2||\theta||_1$. The new term in the cost function is $\langle Bc, c \rangle$, which is linear in $B$, thus it exists a matrix $\mathbf{C_t} = \mathbf{C_t}(c)$ such that $\langle Bc, c \rangle = \langle \mathbf{C_t}, \theta \rangle$. The construction of $\mathbf{C_t}$ is similar to the construction of $\mathbf{X_t}$. The optimization problem (2) then becomes

$$\min_{\theta} ||\mathbf{y} - \mathbf{X_t}\theta||_2^2 + \lambda_1 ||\theta||_1 - \lambda_2 \langle \mathbf{C_t}, \theta \rangle,$$

which can be efficiently solved using any modern $\ell_1$ optimization method (Wright et al., 2009).

## 4 Favoring Motifs in Graphical Models

One of the biggest challenges in bioinformatics is the estimation and understanding of genetic regulatory networks. It has been observed that the structure of these graphs is far from being random: the transcription networks seem to be conformed by a small set of regulation patterns that appear much more often than in random graphs. It is believed that each one of these patterns, called motifs, are responsible of certain specific regulatory functions. Three basic types of motifs are defined (Shen-Orr et al., 2002), the "feedforward loop" being one of the most significant. This motif involves three genes: a regulator X which regulates Y, and a gene Z which is regulated by both X and Y. The representation of these regulations in the network takes the form of a triangle with vertices X, Y, Z.

Although these triangles are very frequent in GRN, the common algorithms discussed in Section 2 seem to fail at producing them. As these models do not consider any topological structure, and the total number of reconstructed triangles is usually much lower than in transcription networks, it seems reasonable to help in the formation of these motifs by favoring the presence of triangles.

In order to move towards a better motif detection, we propose an iterative procedure based on the joint regression model (1). After a first iteration of solving (1), a preliminary symmetric matrix $B$ is obtained. Recall that if $A$ is a graph adjacency matrix, then $A^2$ counts the paths of length 2 between nodes. More specifically, the entry $(i, j)$ of $A^2$ indicates how many paths of length 2 exist from node $i$ to node $j$. Back to the graphical model estimation, this means that if the entry $(B^2)_{ij} \neq 0$ (a length 2 path exists between $i$ and $j$), then by making $B_{ij} \neq 0$ (if it is not already), at least one triangle is added. This suggests that by including weights in the $\ell_1$ penalization, proportionally decreasing with $B^2$, we are favoring those edges that, when added, form a new triangle.

Given the matrix $B$ obtained in the preliminary iteration, we consider the cost matrix $M$ such that $M_{ij} = e^{-\mu(B^2)_{ij}}$, $\mu$ being a positive parameter. This way, if $(B^2)_{ij} = 0$ the weight does not affect the penalty, and if $(B^2)_{ij} \neq 0$, it favors motifs detection. We then solve the optimization problem

$$\min_{B} ||\mathbf{X} - \mathbf{X}B||_F^2 + \lambda_1 ||M \cdot B||_{\ell_1}, \tag{3}$$

where $M \cdot B$ is the pointwise matrix product.

The algorithm iterates between reconstructing the matrix $B$ and updating the weight matrix $M$ (initialized as the identity matrix). Usually after two or three iterations the graph stabilizes.

## 5   Experimental Results

In this section we present numerical and graphical results for the proposed models, and compare them with the original joint regression one.

As discussed in the introduction, there is evidence that most real life networks present scale-free behavior. Therefore, when considering simulated results for validation, we use the model by Barabási & Albert (1999) to generate graphs with this property. Namely, we start from a random graph with $4$ nodes and add one node at a time, randomly connected to one of the existing nodes. The probability of connecting the new node to the node $i$ is proportional to the current degree of node $i$.

Given a graph with adjacency matrix $A$, we simulate the data $\mathbf{X}$ as follows (Liu & Ihler, 2011): let $D$ be a diagonal matrix containing the degree of node $i$ in the entry $D_{ii}$, and consider the matrix $L = \eta D - A$ with $\eta > 1$ so that $L$ is positive definite. We then define the concentration matrix $\Theta = \Lambda^{\frac{1}{2}} L \Lambda^{\frac{1}{2}}$, where $\Lambda$ is the diagonal matrix of $L^{-1}$ (used to normalize the diagonal of $\Sigma = \Theta^{-1}$). Gaussian data $\mathbf{X}$ is then simulated with distribution $\mathcal{N}(0, \Sigma)$. For each algorithm, the parameters are set such that the resulting graph has the same number of edges as the original one. As the total number of edges is then fixed, the false positive (FP) rate can be deduced from the true positive (TP) rate. We therefore report the TP rate only, since it is enough to compare the different performances.

### 5.1   Including Actual Centrality

In this first experiment we show how our model (2) is able to correctly incorporate the prior centrality information, resulting in a more accurate inferred graph, both in detected edges and in topology. The graph of the example in Figure 1 contains 20 nodes. We generated 10 samples and inferred the graph with the joint regression model and with the proposed model (2) using the correct centrality.

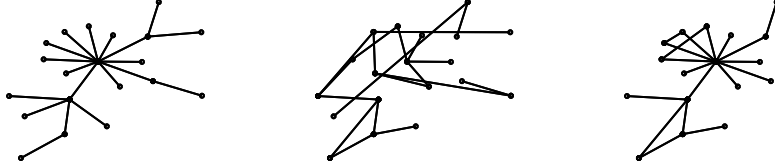

**Figure 1:** Comparison of networks estimated with the simple joint model (1) (middle) and with model (2) (right) using the eigenvector centrality. Original graph on left.

The following more comprehensive test shows the improvement with respect to the basic joint model (1) when the correct centrality is included. For a fixed value of $p = 80$, and for each value of $k$ from 30 to 50, we made 50 runs generating scale-free graphs and simulating data $\mathbf{X}$. From these data we estimated the network with the joint regression model with and without the centrality prior. The TP edge rates in Figure 2(a) are averaged over the 50 runs, and count the correctly detected edges over the (fixed) total number of edges in the network. In addition, Figure 2(b) shows a ROC curve. We generated 300 networks and constructed a ROC curve for each one by varying $\lambda_1$, and we then averaged all the 300 curves. As expected, the incorporation of the known topological property helps in the correct estimation of the graph.

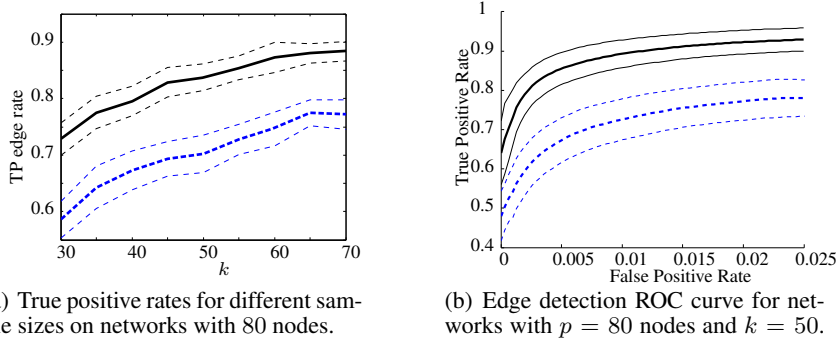

(a) True positive rates for different sample sizes on networks with 80 nodes.

(b) Edge detection ROC curve for networks with $p = 80$ nodes and $k = 50$.

**Figure 2:** Performance comparison of models 2 and 1. In blue (dashed), the standard joint model (1), and in black the proposed model with centrality (2). In thin lines, curves corresponding to $95\%$ confidence intervals.

Following the previous discussion, Figure 3 shows the inner product $\langle v_B, v_C \rangle$ for several runs of model (2), where $v_B$ is the leading eigenvector of the obtained matrix $B$, $C$ is the resulting connectivity matrix (the binarized version of $B$), and $v_C$ its leading eigenvector.

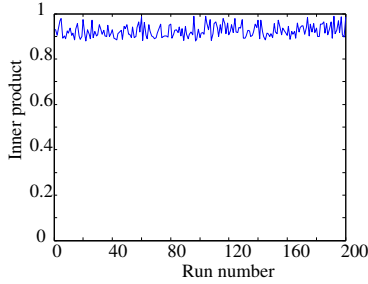

**Figure 3:** Inner product $\langle v_C, v_B \rangle$ for 200 runs.

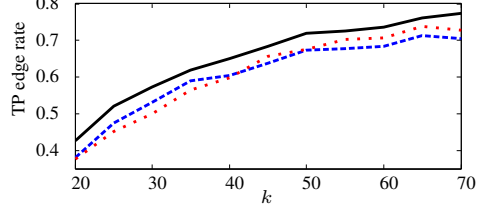

**Figure 4:** True positive edge rates for different sample sizes on a network with 100 nodes. Dashed, the joint model (1), dotted, the PC-Algorithm, and solid the model (2) with centrality estimated from data.

## 5.2  Imposing Centrality Estimated from Data

The previous section shows how the performance of the joint regression model (1) can be improved by incorporating the centrality, when this topology information is available. However, when this vector is unknown, it can be estimated from the data, using an independent algorithm, and then incorporated to the optimization in model (2). We use the PC-Algorithm to estimate the centrality (by computing the dominant eigenvector of the resulting graph), and then we impose it as the vector $c$ in model (2). It turns out that even with a technique not specialized for centrality estimation, this combination outperforms both the joint model (1) and the PC-Algorithm.

We compare the three mentioned models on networks with $p = 100$ nodes for several values of $k$, ranging from 20 to 70. For each value of $k$, we randomly generated ten networks and simulated data $\mathbf{X}$. We then reconstructed the graph using the three techniques and averaged the edge rate over the ten runs. The parameter $\lambda_2$ was obtained via cross validation. Figure 4 shows how the model imposing centrality can improve the other ones without any external information.

## 5.3  Transferring Centrality

In several situations, one may have some information about the topology of the graph to infer, mainly based on other data/graphs known to be similar. For instance, dynamic networks are a good example where one may have some (maybe abundant) old data from the network at a past time $T_1$, some (maybe scarce) new data at time $T_2$, and know that the network topology is similar at the different times. This may be the case of financial, climate, or any time-series data. Outside of temporal varying networks, this topological transference may be useful when we have two graphs of the same kind (say biological networks), which are expected to share some properties, and lots of data is available for the first network but very few samples for the second network are known. We would like to transfer our inferred centrality-based topological knowledge from the first network into the second one, and by that improving the network estimation from limited data.

For these examples, we have an unknown graph $G_1$ corresponding to a $k_1 \times p$ data matrix $\mathbf{X}_1$, which we assume is enough to reasonably estimate $G_1$, and an unknown graph $G_2$ with a $k_2 \times p$ data matrix $\mathbf{X}_2$ (with $k_2 \ll k_1$). Using $\mathbf{X}_2$ only might not be enough to obtain a proper estimate of $G_2$, and considering the whole data together (concatenation of $\mathbf{X}_1$ and $\mathbf{X}_2$) might be an artificial mixture or too strong and lead to basically reconstructing $G_1$. What we really want to do is to transfer some high-level structure of $G_1$ into $G_2$, e.g., just the underlying centrality of $G_1$ is transferred to $G_2$.

In what follows, we show the comparison of inferring the network $G_2$ using only the data $X_2$ in the joint model (1); the concatenation of $X_1$ and $X_2$ in the joint model (1); and finally the centrality estimated from $\mathbf{X}_1$, imposed in model (2), along with data $\mathbf{X}_2$. We fixed the networks size to $p = 100$ and the size of data for $G_1$ to $k_1 = 200$. Given a graph $G_1$, we construct $G_2$ by randomly changing a certain number of edges (32 and 36 edges in Figure 5). For $k_2$ from 35 to 60, we generate data $\mathbf{X}_2$, and we then infer $G_2$ with the methods described above. We averaged over 10 runs.

As it can be observed in Figure 5, the performance of the model including the centrality estimated from $\mathbf{X}_1$ is better than the performance of the classical model, both when using just the data $\mathbf{X}_2$ and the concatenated data $\mathbf{X}_1 | \mathbf{X}_2$. Therefore, we can discard the old data $\mathbf{X}_1$ and keep only the structure (centrality) and still be able to infer a more accurate version of $G_2$.

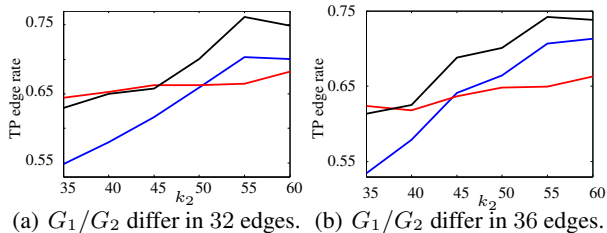

(a) $G_1/G_2$ differ in 32 edges.　(b) $G_1/G_2$ differ in 36 edges.

**Figure 5:** True positive edge rate when estimating the network $G_2$ vs amount of data. In blue, the basic joint model using only $\mathbf{X}_2$, in red using the concatenation of $\mathbf{X}_1$ and $\mathbf{X}_2$, and in black the model (2) using only $\mathbf{X}_2$ with centrality estimated from $\mathbf{X}_1$ as prior.

## 5.4 Experiments on Real Data

### 5.4.1 International Stock Market Data

The stock market is a very complicated system, with lots of time-dependent underlying relationships. In this example we show how the centrality constraint can help to understand these relationships with limited data on times of crisis and times of stability.

We use the daily closing values ($\pi_k$) of some relevant stock market indices from U.S., Canada, Australia, Japan, Hong Kong, U.K., Germany, France, Italy, Switzerland, Netherlands, Austria, Spain, Belgium, Finland, Portugal, Ireland, and Greece. We consider 2 time periods containing a crisis, 5/2007-5/2009 and 5/2009-5/2012, each of which was divided into a "pre-crisis" period, and two more sets (training and testing) covering the actual crisis period. We also consider the relatively stable period 6/1997-6/1999, where the division into these three subsets was made arbitrarily. Using as data the return between two consecutive trading days, defined as $100 \log(\frac{\pi_k}{\pi_{k-1}})$, we first learned the centrality from the "pre-crisis" period, and we then learned three models with the training sets: a classical least-squares regression (LS), the joint regression model (1), and the centrality model (2) with the estimated eigenvector. For each learned model $B$ we computed the "prediction" accuracy $||X_{test} - X_{test}B||_F^2$ in order to evaluate whether the inclusion of the topology improves the estimation. The results are presented in Table 1, illustrating how the topology helps to infer a better model, both in stable and highly changing periods. Additionally, Figure 6 shows a graph learned with the model (2) using the 2009-2012 training data. The discovered relationships make sense, and we can easily identify geographic or socio-economic connections.

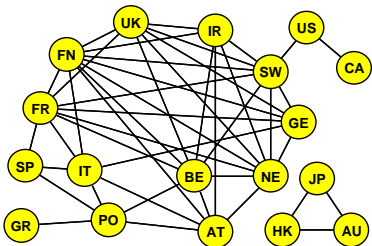

**Figure 6:** Countries network learned with the centrality model.

|  | 97-99 | 07-09 | 09-12 |
|---|---|---|---|
| LS | 2.7 | 3.5 | 14.4 |
| Model (1) | 2.5 | 0.9 | 4.0 |
| Model (2) | **1.9** | **0.6** | **2.4** |

**Table 1:** Mean square error ($\times 10^{-3}$) for the different models.

### 5.4.2 Motif Detection in *Escherichia Coli*

Along this section and the following one, we use as base graph the actual genetic regulation network of the *E. coli*. This graph contains $\approx 400$ nodes, but for practical issues we selected the sub-graph of all nodes with degree $> 1$. This sub-graph $G_E$ contains 186 nodes and 40 feedforward loop motifs.

For the number of samples $k$ varying from 30 to 120, we simulated data $\mathbf{X}$ from $G_E$ and reconstructed the graph using the joint model (1) and the iterative method (3). We then compared the resulting networks to the original one, both in true positive edge rate (recall that this analysis is sufficient since the total number of edges is made constant), and number of motifs correctly detected. The numerical results are shown in Figure 7, where it can be seen that model (3) correctly detect more motifs, with better TP vs FP motif rate, and without detriment of the true positive edge rate.

### 5.4.3 Centrality + Motif Detection

The simplicity of the proposed models allows to combine them with other existing network estimation extensions. We now show the performance of the two models here presented combined (centrality and motifs constraints), tested on the *Escherichia coli* network.

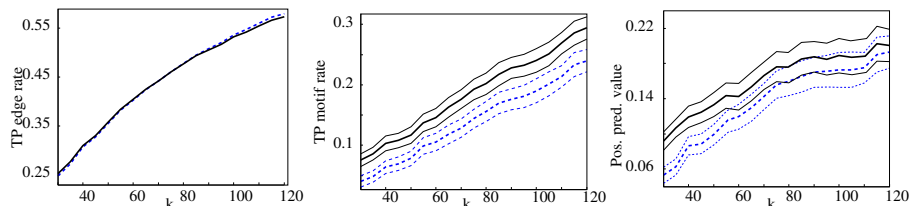

**Figure 7:** Comparison of model (1) (dashed) with proposed model (3) (solid) for the *E. coli* network. Left: TP edge rate. Middle: TP motif rate (motifs correctly detected over the total number of motifs in $G_E$). Right: Positive predictive value (motifs correctly detected over the total number of motifs in the inferred graph).

We first estimate the centrality from the data, as in Section 5.2. Let us assume that we know which ones are the two most central nodes (genes).[1] This information can be used to modify the centrality value for these two nodes, by replacing them by the two highest centrality values typical of scale-free networks (Newman, 2010). For the fixed network $G_E$, we simulated data of different sizes $k$ and reconstructed the graph with the model (1) and with the combination of models (2) and (3). Again, we compared the TP edge rates, the percentage of motifs detected, and the TP/FP motifs rate. Numerical results are shown in Figure 8, where it can be seen that, in addition to the motif detection improvement, now the edge rate is also better. Figure 9 shows the obtained graphs for a specific run.

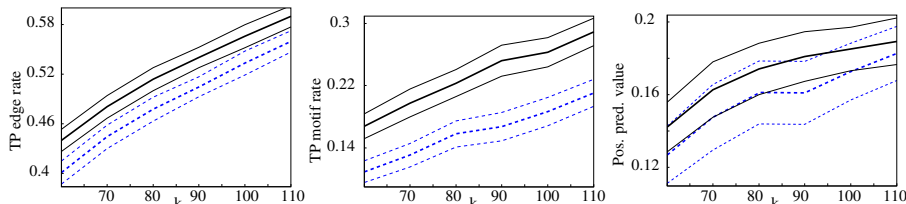

**Figure 8:** Comparison of model (1) (dashed) with the combination of models (2) and (3) (solid) for the *E. coli* network. The combination of the proposed extensions is capable of detecting more motifs while also improving the accuracy of the detected edges. Left: TP edge rate. Middle: TP motif rate. Right: Positive predictive value.

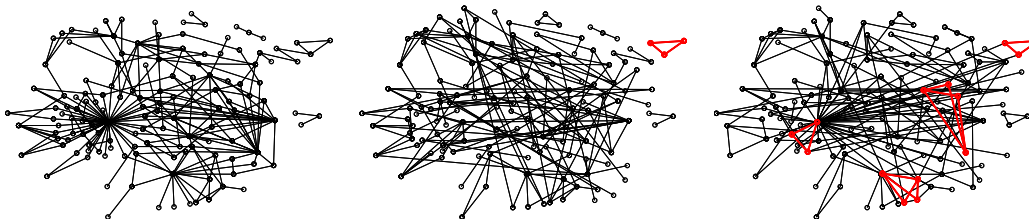

**Figure 9:** Comparison of graphs for the *E. coli* network with $k = 80$. Original network, inferred with model (1) and with the combination of (2) and (3). Note how the combined model is able to better capture the underlying network topology, as quantitative shown in Figure 8. Correctly detected motifs are highlighted.

## 6  Conclusions and Future Work

We proposed two extensions to $\ell_1$ penalized models for precision matrix (network) estimation. The first one incorporates topological information to the optimization, allowing to control the graph centrality. We showed how this model is able to capture the imposed structure when the centrality is provided as prior information, and we also showed how it can improve the performance of the basic joint regression model even when there is no such external information. The second extension favors the appearance of triangles, allowing to better detect motifs in genetic regulatory networks. We combined both models for a better estimation of the *Escherichia coli* GRN.

There are several other graph-topological properties that may provide important information, making it interesting to study which kind of structure can be added to the optimization problem. An algorithm for estimating with high precision the centrality directly from the data would be a great complement to the methods here presented. It is also important to find a model which exploits all the prior information about GRN, including other motifs not explored in this work. Finally, the exploitation of the methods here developed for $\ell_1$-graphs, is the the subject of future research.

## Acknowledgements

Work partially supported by ANII (Uruguay), ONR, NSF, NGA, DARPA, and AFOSR.

## Footnotes

[1]In this case, it is well known that *crp* is the most central node, followed by *fnr*.

## References

Banerjee, O., El Ghaoui, L., and D'Aspremont, A. Model selection through sparse maximum likelihood estimation for multivariate gaussian or binary data. *Journal of Machine Learning Research*, 9:485–516, 2008.

Barabási, A. and Albert, R. Emergence of scaling in random networks. *Science*, 286(5439):509–512, 1999.

Dempster, A. Covariance selection. *Biometrics*, 28(1):157–175, 1972.

Friedman, J., Hastie, T., and Tibshirani, R. Sparse inverse covariance estimation with the graphical lasso. *Biostatistics*, 9(3):432–41, July 2008.

Friedman, J., Hastie, T., and Tibshirani, R. Applications of the lasso and grouped lasso to the estimation of sparse graphical models. Technical report, 2010.

Kalisch, M. and Bühlmann, P. Estimating high-dimensional directed acyclic graphs with the PC-Algorithm. *Journal of Machine Learning Research*, 8:613–636, 2007.

Lauritzen, S. *Graphical Models*. Clarendon Press, Oxford, 1996.

Liu, Q. and Ihler, A. Learning scale free networks by reweighted $\ell_1$ regularization. *AI & Statistics*, 15:40–48, April 2011.

Meinshausen, N. and Bühlmann, P. High-dimensional graphs and variable selection with the Lasso. *The Annals of Statistics*, 34(3):1436–1462, June 2006.

Newman, M. *Networks: An Introduction*. Oxford University Press, Inc., New York, NY, USA, 2010.

Peng, J., Wang, P., Zhou, N., and Zhu, J. Partial correlation estimation by joint sparse regression models. *Journal of the American Statistical Association*, 104(486):735–746, June 2009.

Shen-Orr, S., Milo, R., Mangan, S., and Alon, U. Network motifs in the transcriptional regulation network of Escherichia coli. *Nature Genetics*, 31(1):64–8, May 2002.

Tibshirani, R. Regression shrinkage and selection via the lasso. *Journal of the Royal Statistical Society. Series B*, 58:267–288, 1996.

Wolf, L. and Shashua, A. Feature selection for unsupervised and supervised inference: The emergence of sparsity in a weight-based approach. *Journal of Machine Learning Research*, 6:1855–1887, 2005.

Wright, S., Nowak, R., and Figueiredo, M. Sparse reconstruction by separable approximation. *IEEE Transactions on Signal Processing*, 57(7):2479–2493, 2009.

Yuan, M. and Lin, Y. Model selection and estimation in regression with grouped variables. *Journal of the Royal Statistical Society: Series B*, 68(1):49–67, 2006.

Yuan, M. and Lin, Y. Model selection and estimation in the Gaussian graphical model. *Biometrika*, 94(1):19–35, February 2007.

Zeitouni, O. Personal communication, 2012.

